# Sex with Support Vector Machines

**Baback Moghaddam**
Mitsubishi Electric Research Laboratory
Cambridge MA 02139, USA
baback@merl.com

**Ming-Hsuan Yang**
University of Illinois at Urbana-Champaign
Urbana, IL 61801 USA
mhyang@vision.ai.uiuc.edu

## Abstract

Nonlinear Support Vector Machines (SVMs) are investigated for visual sex classification with low resolution "thumbnail" faces (21-by-12 pixels) processed from 1,755 images from the FERET face database. The performance of SVMs is shown to be superior to traditional pattern classifiers (Linear, Quadratic, Fisher Linear Discriminant, Nearest-Neighbor) as well as more modern techniques such as Radial Basis Function (RBF) classifiers and large ensemble-RBF networks. Furthermore, the SVM performance (3.4% error) is currently the best result reported in the open literature.

## 1 Introduction

In recent years, SVMs have been successfully applied to various tasks in computational face-processing. These include face detection [14], face pose discrimination [12] and face recognition [16]. Although facial sex classification has attracted much attention in the psychological literature [1, 4, 8, 15], relatively few computatinal learning methods have been proposed. We will briefly review and summarize the prior art in facial sex classification.[1]

Gollomb *et al.* [10] trained a fully connected two-layer neural network, SEXNET, to identify sex from 30-by-30 face images. Their experiments on a set of 90 photos (45 males and 45 females) gave an average error rate of 8.1% compared to an average error rate of 11.6% from a study of five human subjects. Brunelli and Poggio [2]

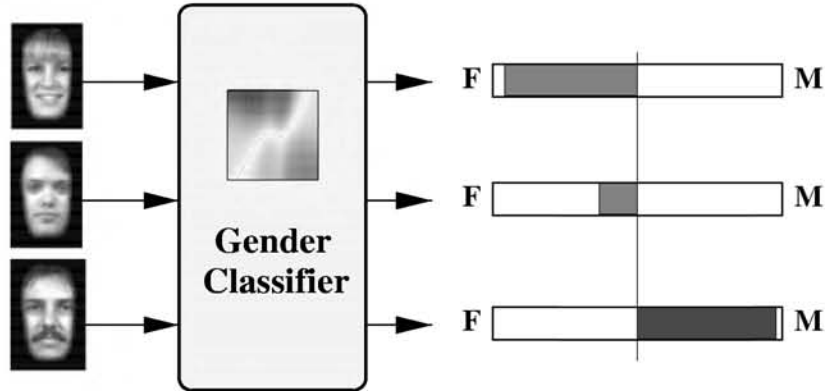

Figure 1: Sex classifier

developed HyperBF networks for sex classification in which two competing RBF networks, one for male and the other for female, were trained using 16 geometric features as inputs (*e.g.*, pupil to eyebrow separation, eyebrow thickness, and nose width). The results on a data set of 168 images (21 males and 21 females) show an average error rate of 21%. Using similar techniques as Golomb *et al.* [10] and Cottrell and Metcalfe [6], Tamura *et al.* [18] used multi-layer neural networks to classify sex from face images at multiple resolutions (from 32-by-32 to 8-by-8 pixels). Their experiments on 30 test images show that their network was able to determine sex from 8-by-8 images with an average error rate of 7%. Instead of using a vector of gray levels to represent faces, Wiskott *et al.* [20] used labeled graphs of two-dimensional views to describe faces. The nodes were represented by wavelet-based local "jets" and the edges were labeled with distance vectors similar to the geometric features in [3]. They used a small set of controlled model graphs of males and females to encode "general face knowledge," in order to generate graphs of new faces by elastic graph matching. For each new face, a composite reconstruction was generated using the nodes in the model graphs. The sex of the majority of nodes used in the composite graph was used for classification. The error rate of their experiments on a gallery of 112 face images was 9.8%. Recently, Gutta *et al.* [11] proposed a hybrid classifier based on neural networks (RBFs) and inductive decision trees with Quinlan's C4.5 algorithm. Experiments with 3000 FERET faces of size 64-by-72 pixels yielded an error rate of 4%.

## 2 Sex Classifiers

A generic sex classifier is shown in Figure 1. An input facial image $\mathbf{x}$ generates a scalar output $f(\mathbf{x})$ whose polarity – sign of $f(\mathbf{x})$ – determines class membership. The magnitude $\|f(\mathbf{x})\|$ can usually be interpreted as a measure of belief or certainty in the decision made. Nearly all binary classifiers can be viewed in these terms; for density-based classifiers (Linear, Quadratic and Fisher) the output function $f(\mathbf{x})$ is a log likelihood ratio, whereas for kernel-based classifiers (Nearest-Neighbor, RBFs and SVMs) the output is a "potential field" related to the distance from the separating boundary.

## 2.1 Support Vector Machines

A Support Vector Machine is a learning algorithm for pattern classification and regression [19, 5]. The basic training principle behind SVMs is finding the optimal linear hyperplane such that the expected classification error for unseen test samples is minimized — *i.e.*, good generalization performance. According to the structural risk minimization inductive principle [19], a function that classifies the training data accurately and which belongs to a set of functions with the lowest VC dimension [5] will generalize best regardless of the dimensionality of the input space. Based on this principle, a linear SVM uses a systematic approach to finding a class of functions with the lowest VC dimension. For linearly non-separable data, SVMs can (nonlinearly) map the input to a high dimensional feature space where a linear hyperplane can be found. Although there is no guarantee that a linear separable solution will always exist in the high dimensional space, in practice it is quite feasible to construct a working solution.

Given a labeled set of $M$ training samples $(\mathbf{x}_i, y_i)$, where $\mathbf{x}_i \in R^N$ and $y_i$ is the associated label ($y_i \in \{-1, 1\}$), a SVM classifier finds the optimal hyperplane that correctly separates (classifies) the data points while maximizing the distance of either class from the hyperplane (the margin). Vapnik [19] shows that maximizing the margin distance is equivalent to minimizing the VC dimension in constructing an optimal hyperplane. Computing the best hyperplane is posed as a constrained optimization problem and solved using quadratic programming techniques. The discriminant hyperplane is defined by the level set of

$$f(\mathbf{x}) = \sum_{i=1}^{M} y_i \, \alpha_i \cdot k(\mathbf{x}, \mathbf{x}_i) + b$$

where $k(\cdot, \cdot)$ is a kernel function and the sign of $f(\mathbf{x})$ determines the membership of $\mathbf{x}$. Constructing an optimal hyperplane is equivalent to finding all the nonzero $\alpha_i$. Any vector $\mathbf{x}_i$ that corresponds to a nonzero $\alpha_i$ is a *supported vector* (SV) of the optimal hyperplane. A desirable feature of SVMs is that the number of training points which are retained as support vectors is usually quite small, thus providing a compact classifier.

For a linear SVM, the kernel function is just a simple dot product in the input space while the kernel function in a nonlinear SVM effectively projects the samples to a feature space of higher (possibly infinite) dimension via a nonlinear mapping function:

$$\Phi : R^N \to F^M, \quad M \gg N$$

and then constructs a hyperplane in $F$. The motivation behind this mapping is that it makes possible a larger class of discriminative functions with which to find a linear hyperplane in the high dimensional feature space. Using Mercer's theorem [7], the expensive calculations required in projecting samples into the high dimensional feature space can be replaced by a much simpler kernel function satisfying the condition

$$k(\mathbf{x}, \mathbf{x}_i) = \Phi(\mathbf{x}) \cdot \Phi(\mathbf{x}_i)$$

where $\Phi$ is the implicit nonlinear projection. Several kernel functions, such as polynomials and radial basis functions, have been shown to satisfy Mercer's theorem

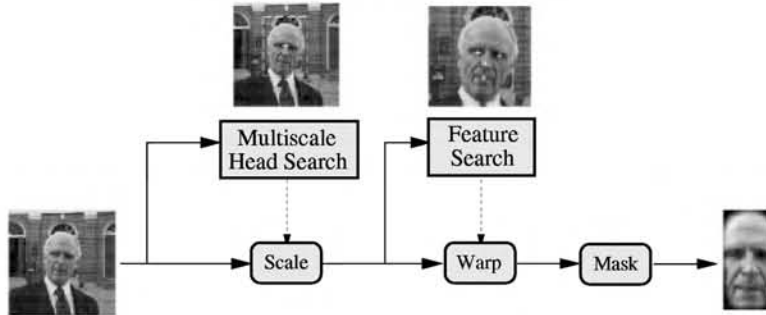

Figure 2: Automatic face alignment system [13].

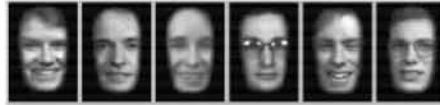

Figure 3: Some processed FERET faces, at high resolution

and have been used successfully in nonlinear SVMs. In fact, by using different kernel functions, SVMs can implement a variety of learning machines, some of which coincide with classical architectures. Nevertheless, automatic selection of the "right" kernel function and its associated parameters remains problematic and in practice one must resort to trial and error for model selection.

A radial basis function (RBF) network is also a kernel-based technique for improved generalization, but it is based instead on regularization theory [17]. The number of radial bases and their centers in a conventional RBF network is predetermined, often by $k$−means clustering. In contrast, a SVM with the same RBF kernel will automatically determine the number and location of the centers, as well as the weights and thresholds that minimize an upper bound on the expected risk. Recently, Evgeniou *et al.* [9] have shown that both SVMs and RBFs can be formulated under a unified framework in the context of Vapnik's theory of statistical learning [19]. As such, SVMs provide a more systematic approach to classification than classical RBF and various other neural networks.

## 3 Experiments

In our study, 256-by-384 pixel FERET "mug-shots" were pre-processed using an automatic face-processing system which compensates for translation, scale, as well as slight rotations. Shown in Figure 2, this system is described in detail in [13] and uses maximum-likelihood estimation for face detection, affine warping for geometric shape alignment and contrast normalization for ambient lighting variations. The resulting output "face-prints" in Figure 2 were standardized to 80-by-40 (full) resolution. These "face-prints" were further sub-sampled to 21-by-12 pixel "thumbnails" for our low resolution experiments. Figure 3 shows a few examples of processed face-prints (note that these faces contain little or no hair information). A total of 1755 thumbnails (1044 males and 711 females) were used

| Classifier | Error Rate | | |
|---|---|---|---|
| | Overall | Male | Female |
| **SVM with Gaussian RBF kernel** | **3.38%** | **2.05%** | **4.79%** |
| **SVM with Cubic polynomial kernel** | **4.88%** | **4.21%** | **5.59%** |
| Large ensemble-RBF | 5.54% | 4.59% | 6.55% |
| Classical RBF | 7.79% | 6.89% | 8.75% |
| Quadratic classifier | 10.63% | 9.44% | 11.88% |
| Fisher linear discriminant | 13.03% | 12.31% | 13.78% |
| Nearest neighbor | 27.16% | 26.53% | 28.04% |
| Linear classifier | 58.95% | 58.47% | 59.45% |

Table 1: Experimental results with thumbnails.

in our experiments. For each classifier, the average error rate was estimated with 5-fold cross validation (CV) — *i.e.*, a 5-way dataset split, with 4/5th used for training and 1/5th used for testing, with 4 subsequent non-overlapping rotations. The average size of the training set was 1496 (793 males and 713 females) and the average size of the test set was 259 (133 males and 126 females).

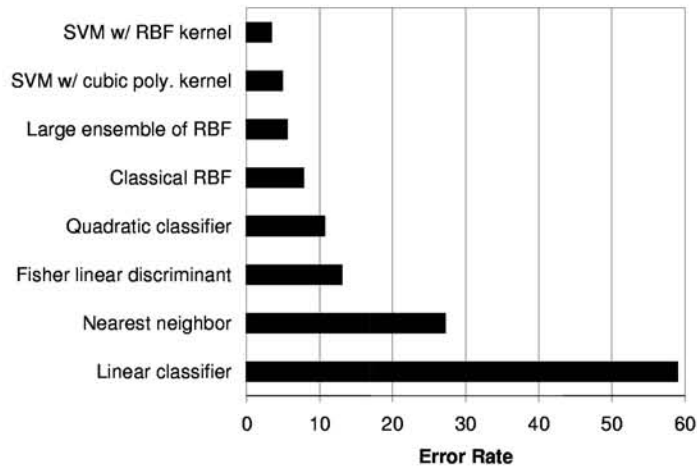

Figure 4: Error rates of various classifiers

The SVM classifier was first tested with various kernels in order to explore the space of models and performance. A Gaussian RBF kernel was found to perform the best (in terms of error rate), followed by a cubic polynomial kernel as second best. In the large ensemble-RBF experiment, the number of radial bases was incremented until the error fell below a set threshold. The average number of radial bases in the large ensemble-RBF was found to be 1289 which corresponds to 86% of the training set. The number of radial bases for classical RBF networks was heuristically set to 20

prior to actual training and testing. Quadratic, Linear and Fisher classifiers were implemented using Gaussian distributions and in each case a likelihood ratio test was used for classification. The average error rates of all the classifiers tested with 21-by-12 pixel thumbnails are reported in Table 1 and summarized in Figure 3.

The SVMs out-performed all other classifiers, although the performance of large ensemble-RBF networks was close to SVMs. However, nearly 90% of the training set was retained as radial bases by the large ensemble-RBF. In contrast, the number of support vectors found by both SVMs was only about 20% of the training set. We also applied SVMs to classification based on high resolution images. The Gaussian and cubic kernel performed equally well at both low and high resolutions with only a slight 1% error rate difference. We note that as indicated in Table 1, all the classifiers had higher error rates in classifying females, most likely due to the general lack of prominent and distinct facial features in female faces, as compared to males.

## 4   Discussion

We have presented a comprehensive evaluation of various classification methods for determination of sex from facial images. The non-triviality of this task (made even harder by our "hairless" low resolution faces) is demonstrated by the fact that a linear classifier had an error rate of 60% (*i.e.*, worse than a random coin flip). Furthermore, an acceptable error rate ($< 5\%$) for the large ensemble-RBF network required storage of 86% of the training set (SVMs required about 20%). Storage of the entire dataset in the form of the nearest-neighbor classifier yielded too high an error rate (30%). Clearly, SVMs succeeded in the difficult task of finding a near-optimal class partition in face space with the added economy of a small number of support faces.

Given the relative success of previous studies with low resolution faces it is re-assuring that 21-by-12 faces can, in fact, be used for reliable sex classification. However, most of the previous studies used datasets of relatively few faces, consequently with little statistical significance in the reported results. The most directly comparable study to ours is that of Gutta *et al.* [11], which also used FERET faces. With a dataset of 3000 faces at a resolution of 64-by-72, their hybrid RBF/Decision-Tree classifier achieved a 4% error rate. In our study, with 1800 faces at a resolution of 21-by-12, a Gaussian kernel SVM was able to achieve a 3.4% error rate. Both studies use extensive cross validation to estimate the error rates. Given our results with SVMs, it is clear that better performance at even lower resolutions is possible with this learning technique.

## Footnotes

[1]Sex classification is also referred to as gender classification (for political correctness). However, given the two distinct biological classes, the scientifically correct term is *sex* classification. Gender often denotes a fuzzy continuum of feminine $\longleftrightarrow$ masculine [1].

## References

[1] V. Bruce, A. M. Burton, N. Dench, E. Hanna, P. Healey, O. Mason, A. Coombes, R. Fright, and A. Linney. Sex discrimination: How do we tell the difference between male and female faces? *Perception*, 22:131–152, 1993.

[2] R. Brunelli and T. Poggio. Hyperbf networks for gender classification. In *Proceedings of the DARPA Image Understanding Workshop*, pages 311–314, 1992.

[3] R. Brunelli and T. Poggio. Face recognition : Features vs. templates. *IEEE Transactions on Pattern Analysis and Machine Intelligence*, 15(10), October 1993.

[4] A. M. Burton, V. Bruce, and N. Dench. What's the difference between men and women? evidence from facial measurement. *Perception*, 22:153–176, 1993.

[5] C. Cortes and V. Vapnik. Support vector networks. *Machine Learning*, 20, 1995.

[6] Garrison W. Cottrell. Empath: Face, emotion, and gender recognition using holons. In *Advances in Neural Information Processing Systems*, pages 564–571, 1991.

[7] R. Courant and D. Hilbert. *Methods of Mathematical Physiacs*, volume 1. Interscience, New-York, 1953.

[8] B. Edelman, D. Valentin, and H. Abdi. Sex classification of face areas: how well can a linear neural network predict human performance. *Journal of Biological System*, 6(3):241–264, 1998.

[9] Theodoros Evgeniou, Massimiliano Pontil, and Tomaso Poggio. A unified framework for regularization networks and support vector machines. Technical Report AI Memo No. 1654, MIT, 1999.

[10] B. A. Golomb, D. T. Lawrence, and T. J. Sejnowski. Sexnet: A neural network identifies sex from human faces. In *Advances in Neural Information Processing Systems*, pages 572–577, 1991.

[11] S. Gutta, H. Wechsler, and P. J. Phillips. Gender and ethnic classification. In *Proceedings of the IEEE International Automatic Face and Gesture Recognition*, pages 194–199, 1998.

[12] J. Huang, X. Shao, and H. Wechsler. Face pose discrimination using support vector machines. In *Proc. of 14th Int'l Conf. on Pattern Recognition (ICPR'98)*, pages 154–156, August 1998.

[13] B. Moghaddam and A. Pentland. Probabilistic visual learning for object representation. *IEEE Transactions on Pattern Analysis and Machine Intelligence*, PAMI-19(7):696–710, July 1997.

[14] E. Osuna, R. Freund, and F. Girosi. Training support vector machines: an application to face detection. In *Proceedings of the IEEE Computer Society Conference on Computer Vision and Pattern Recognition*, pages 130–136, 1997.

[15] A. J. O'Toole, T. Vetter, N. F. Troje, and H. H. Bulthoff. Sex classification is better with three-dimensional structure than with image intensity information. *Perception*, 26:75–84, 1997.

[16] P. J. Phillips. Support vector machines applied to face recognition. In M. S. Kearns, S. Solla, and D. Cohen, editors, *Advances in Neural Information Processing Systems 11*, volume 11, pages 803–809. MIT Press, 1998.

[17] T. Poggio and F. Girosi. Networks for approximation and learning. *Proceedings of the IEEE*, 78(9):1481–1497, 1990.

[18] S. Tamura, H. Kawai, and H. Mitsumoto. Male/female identification from 8 × 6 very low resolution face images by neural network. *Pattern Recognition*, 29(2):331–335, 1996.

[19] V. Vapnik. *The Nature of Statistical Learning Theory*. Springer, 1995.

[20] Laurenz Wiskott, Jean-Marc Fellous, Norbert Krüger, and Christoph von der Malsburg. Face recognition and gender determination. In *Proceedings of the International Workshop on Automatic Face and Gesture Recognition*, pages 92–97, 1995.
